# BACKPROPAGATION AND ITS APPLICATION TO HANDWRITTEN SIGNATURE VERIFICATION

Dorothy A. Mighell
Electrical Eng. Dept.
Info. Systems Lab
Stanford University
Stanford, CA 94305

Timothy S. Wilkinson
Electrical Eng. Dept.
Info. Systems Lab
Stanford University
Stanford, CA 94305

Joseph W. Goodman
Electrical Eng. Dept.
Info. Systems Lab
Stanford University
Stanford, CA 94305

## ABSTRACT

A pool of handwritten signatures is used to train a neural network for the task of deciding whether or not a given signature is a forgery. The network is a feedforward net, with a binary image as input. There is a hidden layer, with a single unit output layer. The weights are adjusted according to the backpropagation algorithm. The signatures are entered into a C software program through the use of a Datacopy Electronic Digitizing Camera. The binary signatures are normalized and centered. The performance is examined as a function of the training set and network structure. The best scores are on the order of 2% true signature rejection with 2-4% false signature acceptance.

## INTRODUCTION

Signatures are used everyday to authorize the transfer of funds for millions of people. We use our signature as a form of identity, consent, and authorization. Bank checks, credit cards, legal documents and waivers all require the everchanging personalized signature. Forgeries on such transactions amount to millions of dollars lost each year. A trained eye can spot most forgeries, but it is not cost effective to handcheck all signatures due to the massive number of daily transactions. Consequently, only disputed claims and checks written for large amounts are verified. The consumer would certainly benefit from the added protection of automated verification. Neural networks lend themselves very well to signature verification. Previously, they have proven applicable to other signal processing tasks, such as character recognition {Fukishima, 1986} {Jackel, 1988}, sonar target classification {Gorman, 1986}, and control - as in the broom balancer {Tolat, 1988}.

## HANDWRITING ANALYSIS

Signature verification is only one aspect of the study of handwriting analysis. Recognition is the objective, whether it be of the writer or the characters. Writer recognition can be further broken down into identification and verification. Identi-

fication selects the author of a sample from among a group of writers. Verification confirms or rejects a written sample for a single author. In both cases, it is the style of writing that is important.

Deciphering written text is the basis of character recognition. In this task, linguistic information such as the individual characters or words are extracted from the text. Style must be eliminated to get at the content. A very important application of character recognition is automated reading of zip-codes in the post office {Jackel, 1988}.

Data for handwriting analysis may be either dynamic or static. Dynamic data requires special devices for capturing the temporal characteristics of the sample. Features such as pressure, velocity, and position are examined in the dynamic framework. Such analysis is usually performed on-line in real time.

Static analysis uses the final trace of the writing, as it appears on paper. Static analysis does not require any special processing devices while the signature is being produced. Centralized verification becomes possible, and the processing may be done off-line.

Work has been done in both static and dynamic analysis {Sato, 1982} {Nemcek, 1974}. Generally, signature verification efforts have been more successful using the dynamic information. It would be extremely useful though, to perform the verification using only the written signature. This would eliminate the need for costly machinery at every place of business. Personal checks may also be verified through a static signature analysis.

## TASK

The handwriting analysis task with which this paper is concerned is that of signature verification using an off-line method to detect casual forgeries. Casual forgeries are non-professional forgeries, in which the writer does not practice reproducing the signature. The writer may not even have a copy of the true signature. Casual forgeries are very important to detect. They are far more abundant, and involve greater monetary losses than professional forgeries. This signature verification task falls into the writer recognition category, in which the style of writing is the important variable. The off-line analysis allows centralized verification at a lower cost and broader use.

## HANDWRITTEN SIGNATURES

The signatures for this project were gathered from individuals to produce a pool of 80 true signatures and 66 forgeries. These are signatures, true and false, for one person. There is a further collection of signatures, both true and false, for other persons, but the majority of the results presented will be for the one individual. It will be clear when other individuals are included in the demonstration.

The signatures are collected on 3x5 index cards which have a small blue box as

a guideline. The cards are scanned with a CCD array camera from Datacopy, and thresholded to produce binary images. These binary images are centered and normalized to fit into a 128x64 matrix. Either the entire 128x64 image is presented as input, or a 90x64 image of the three initials alone is presented. It is also possible to present preprocessed inputs to the network.

## SOFTWARE SIMULATION

The type of learning algorithm employed is that of backpropagation. Both dwell and momentum are included. Dwell is the type of scheduling employed, in which an image is presented to the network, and the network is allowed to "dwell" on that input for a few iterations while updating its weights. C. Rosenberg and T. Sejnowski have done a few studies on the effects of scheduling on learning {Rosenberg, 1986}. Momentum is a term included in the change of weights equation to speed up learning {Rumelhart, 1986}.

The software is written in Microsoft C, and run on an IBM PC/AT with an 80287 math co-processor chip.

Included in the simulation is a piece-wise linear approximation to the sigmoid transfer function as shown in Figure 1. This greatly improves the speed of calculation, because an exponential is not calculated. The non-linearity is kept to allow for layering of the network. Most of the details of initialization and update are the same as that reported in NetTalk {Sejnowski, 1986}.

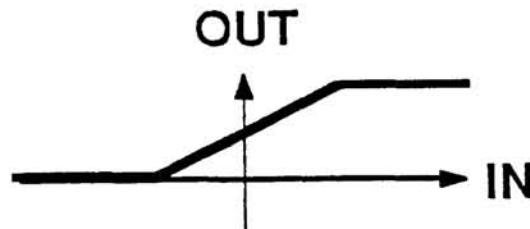

**Figure 1.** Piece-wise linear transfer function.

Many different nets were trained in this signature verification project, all of which were feed-forward. The output layer most often consisted of a single output neuron, but 5 output neurons have been used as well. If a hidden layer was used, then the number of hidden units ranged from 2 to 53. The networks were both fully-connected and partially-connected.

## SAMPLE RUN

The simplest network is that of a single neuron taking all 128x64 pixels as input, plus one bias. Each pixel has a weight associated with it, so that the total number of weights is 128x64 + 1 = 8193. Each white pixel is assigned an input value of +1, each black pixel has a value of -1. The training set consists of 10 true signatures

with 10 forgeries. Figure 2a depicts the network structure of this sample run.

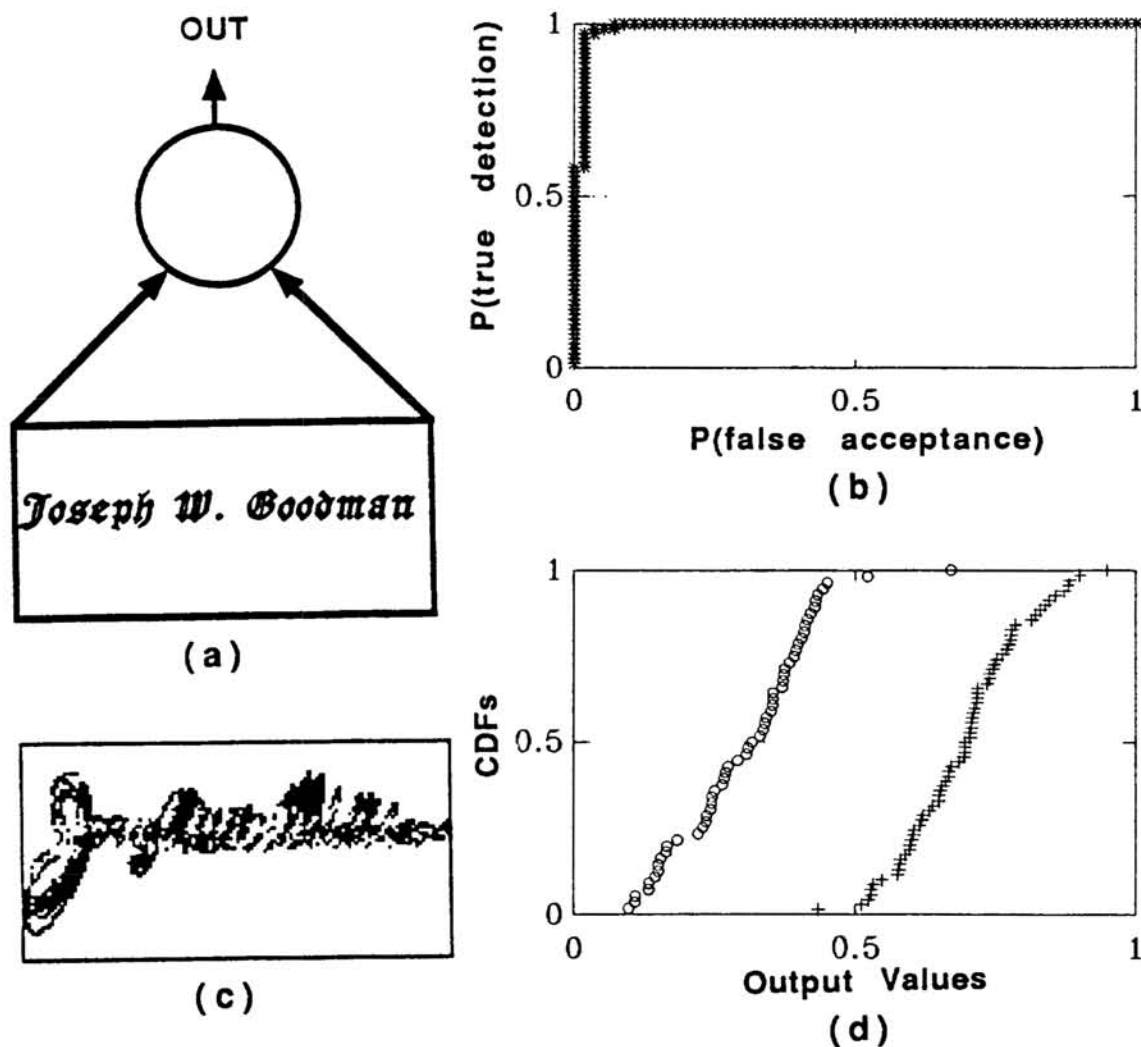

**Figure 2.** Sample run.

a) Network = one output neuron, one weight per pixel, fully connected. Training set = 10 true signatures + 10 forgeries.

b) ROC plot for the sample run. (Probability of false acceptance vs probability of true detection). Test set = 70 true signatures + 56 forgeries.

c) Clipped picture of the weights for the sample run. White = positive weight, black = negative weight.

d) Cumulative distribution function for the true signatures (+) and for the forgeries (o) of the sample run.

The network is trained on these 20 signatures until all signatures are classified

correctly. The trained network is then tested on the remaining 70 true signatures and 56 forgeries.

The results are depicted in Figures 2b and 2d. Figure 2b is a radar operating characteristic curve, or roc plot for short. In this presentation of data, the probability of detecting a true signature is plotted against the probability of accepting a forgery. Roc plots have been used for some time in the radar sciences as a means for visualizing performance {Marcum, 1960}. A perfect roc plot has a right angle in the upper left-hand corner which would show perfect separation of true signatures from forgeries. The curve is plotted by varying the threshold for classification. Everything above the threshold is labeled a true signature, everything below the threshold is labeled a forgery. The roc plot in Figure 2b is close to perfect, but there is some overlap in the output values of the true signatures and forgeries. The overlap can be seen in the cumulative distribution functions (cdfs) for the true and false signatures as shown in Figure 2d. As seen in the cdfs, there is fairly good separation of the output values. For a given threshold of 0.5, the network produces 1% rejection of true signatures as false, with 4% acceptance of forgeries as being true. If one lowers the threshold for classification down to 0.43, the true rejection becomes nil, with a false acceptance of 7% . A simplified picture of the weights is shown in Figure 2c, with white pixels designating positive weights, and black pixels negative weights.

## OTHER NETWORKS

The sample run above was expanded to include 2 and 3 hidden neurons with the single output neuron. The results were similar to the single unit network, implying that the separation is linear.

The 128x64 input image was also divided into regions, with each region feeding into a single neuron. In one network structure, the input was sectioned into 32 equally sized regions of 16x16 pixels. The hidden layer thus has 32 neurons, each neuron receiving 16x16 + 1 inputs. The output neuron had 33 inputs. Likewise, the input image was divided into 53 regions of 16x16 pixels, this time overlapping.

Finally, only the initials were presented to the network. (Handwriting experts have noted that leading strokes and separate capital letters are very significant in classification {Osborn, 1929}.) In this case, two types of networks were devised. The first had a single output neuron, the second had three hidden neurons plus one output neuron. Each of the hidden neurons received inputs from only one initial, rather than from all three. The network with the single output neuron produced the best results of all, with 2% true rejection and 2% false acceptance.

## IMPORTANCE OF FORGERIES IN THE TRAINING SET

In all cases, the networks performed much better when forgeries were included in the training set. When an all-white image is presented as the only forgery, performance deteriorates significantly. When no forgeries are present, the network decides that

all signatures are true signatures. It is therefore desirable to include actual forgeries in the training set, yet they may be impractical to obtain. One possibility for avoiding the collection of forgeries is to use computer generated forgeries. Another is to distort the true signatures. A third is to use true signatures of other people as forgeries for the person in question. The attraction of this last option is that the masquerading forgeries are already available for use.

## NETWORK WITHOUT FORGERIES

To test the use of true signatures of other people for forgeries, the following network is devised. Once again, the input is the 128x64 pixel image. The output layer is comprised of five output neurons fully connected to the input image. The function of each output neuron is to be active when presented with a particular persons' signature. When a forgery is present, the output is to be low. Figure 3a depicts this network. The training set has 50 true signatures, ten for each of five people. Each signature has a desired output of true for one neuron, and false for the remaining four neurons. Once the network is trained, it is tested on 210 true signatures and 150 forgeries. Figures 3b and 3c record the results. At a threshold of 0.5, the true rejection is 3% and the false acceptance is 14%. Decreasing the threshold down to 0.41 gives 0% true rejection and 28% false acceptance. These results are similar to the sample run, though not as good. This is a simple demonstration of the use of other true signatures as forgeries. More sophisticated techniques could improve the discrimination. For instance, selecting names with similar lengths or spelling should improve the classification.

## CONCLUSION

Automated signature verification systems would be extremely important in the business world for verifying monetary transactions. Countless dollars are lost each day to instances of casual forgeries. An artificial neural network employing the backpropagation learning algorithm has been trained on both true and false signatures for classification. The results have been very good: 2% rejection of genuine signatures with 2% acceptance of forgeries. The analysis requires only the static picture of the signature, thereby offering widespread use through centralized verification. True signatures of other people may substitute for the forgeries in the training set - eliminating the need for collecting non-genuine signatures.

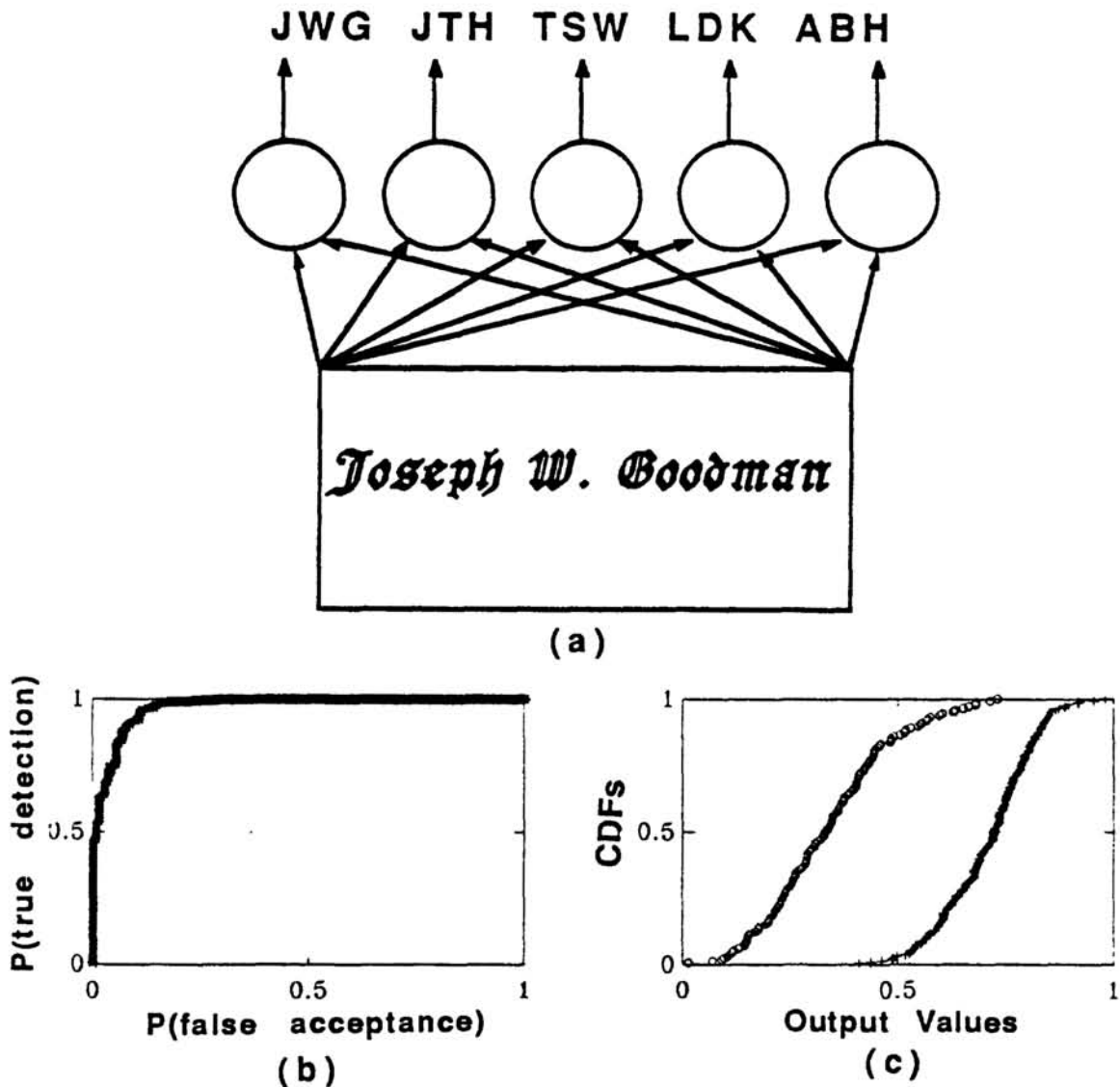

**(a)**

**(b)**

**(c)**

**Figure 3.** Network without forgeries for 5 individuals.

a) Network = 5 output neurons, one for each individual, as indicated by the initials. Training set = 10 true signatures for each individual.

b) ROC plot for the network without forgeries.
Test set = 210 true signatures + 150 forgeries.

c) Cumulative distribution function for the true signatures (+) and for the forgeries (o) of the network without forgeries.

## References

K. Fukishima and S. Miyake, "Neocognitron: A biocybernetic approach to visual pattern recognition", in *NHK Laboratories Note*, Vol. 336, Sep 1986 (NHK Science and Technical Research Laboratories, Tokyo).

P. Gorman and T. J. Sejnowski, "Learned classification of sonar targets using a massively parallel network", in the proceedings of the IEEE ASSP Oct 21, 1986 DSP Workshop, Chatham, MA.

L. D. Jackel, H. P. Graf, W. Hubbard, J. S. Denker, and D. Henderson, "An application of neural net chips: handwritten digit recognition", in *IEEE International Conference on Neural Networks 1988*, II 107-115.

J. T. Marcum, "A statistical theory of target detection by pulsed radar", in *IRE Transactions in Information Theory*, Vol. IT-6 (Apr.), pp 145-267, 1960.

W. F. Nemcek and W. C. Lin, "Experimental investigation of automatic signature verification" in *IEEE Transactions on Systems, Man, and Cybernetics*, Jan. 1974, pp 121-126.

A. S. Osborn, *Questioned Documents*, 2nd edition (Boyd Printing Co, Albany NY) 1929.

C. R. Rosenberg and T. J. Sejnowski, "The spacing effect on NETtalk, a massively parallel network", in *Proceedings of the Eighth Annual Conference of the Cognitive Science Society*, (Hillsdale, New Jersey: Lawrence Erlbaum Associates, 1986) 72-89.

D. E. Rumelhart, G. E. Hinton, and R. J. Williams, "Learning internal representations by error propagation", in *Parallel Distributed Processing: Explorations in the Microstructures of Cognition. Vol. 1: Foundations*, edited by D. E. Rumelhart & J. L. McClelland, (MIT Press, 1986).

Y. Sato and K. Kogure, "Online signature verification based on shape, motion, and writing pressure", in *Proceedings of the 6th International Conference on Pattern Recognition*, Vol. 2, pp 823-826 (IEEE NY) 1982.

T. J. Sejnowski and C. R. Rosenberg, "NETtalk: A Parallel Network that Learns to Read Aloud", Johns Hopkins University Department of Electrical Engineering and Computer Science Technical Report JHU/EECS-86/01, (1986).

V. V. Tolat and B. Widrow, "An adaptive 'broom balancer' with visual inputs", in *IEEE International Conference on Neural Networks 1988*, II 641-647.
